# Learning Gaussian Process Kernels via Hierarchical Bayes

**Anton Schwaighofer**
Fraunhofer FIRST
Intelligent Data Analysis (IDA)
Kekuléstrasse 7, 12489 Berlin
anton@first.fhg.de

**Volker Tresp, Kai Yu**
Siemens Corporate Technology
Information and Communications
81730 Munich, Germany
{volker.tresp,kai.yu}@siemens.com

## Abstract

We present a novel method for learning with Gaussian process regression in a hierarchical Bayesian framework. In a first step, kernel matrices on a fixed set of input points are learned from data using a simple and efficient EM algorithm. This step is nonparametric, in that it does not require a parametric form of covariance function. In a second step, kernel functions are fitted to approximate the learned covariance matrix using a generalized Nyström method, which results in a complex, data driven kernel. We evaluate our approach as a recommendation engine for art images, where the proposed hierarchical Bayesian method leads to excellent prediction performance.

## 1 Introduction

In many real-world application domains, the available training data sets are quite small, which makes learning and model selection difficult. For example, in the user preference modelling problem we will consider later, learning a preference model would amount to fitting a model based on only 20 samples of a user's preference data. Fortunately, there are situations where individual data sets are small, but data from similar scenarios can be obtained. Returning to the example of preference modelling, data for many different users are typically available. This data stems from clearly separate individuals, but we can expect that models can borrow strength from data of users with similar tastes. Typically, such problems have been handled by either mixed effects models or hierarchical Bayesian modelling.

In this paper we present a novel approach to hierarchical Bayesian modelling in the context of Gaussian process regression, with an application to recommender systems. Here, hierarchical Bayesian modelling essentially means to learn the mean and covariance function of the Gaussian process.

In a first step, a common collaborative kernel matrix is learned from the data via a simple and efficient EM algorithm. This circumvents the problem of kernel design, as no parametric form of kernel function is required here. Thus, this form of learning a covariance matrix is also suited for problems with complex covariance structure (e.g. nonstationarity).

A portion of the learned covariance matrix can be explained by the input features and, thus,

generalized to new objects via a content-based kernel smoother. Thus, in a second step, we generalize the covariance matrix (learned by the EM-algorithm) to new items using a generalized Nyström method. The result is a complex content-based kernel which itself is a weighted superposition of simple smoothing kernels. This second part could also be applied to other situations where one needs to extrapolate a covariance matrix on a finite set (e.g. a graph) to a continuous input space, as, for example, required in induction for semi-supervised learning [14].

The paper is organized as follows. Sec. 2 casts Gaussian process regression in a hierarchical Bayesian framework, and shows the EM updates to learn the covariance matrix in the first step. Extrapolating the covariance matrix is shown in Sec. 3. We illustrate the function of the EM-learning on a toy example in Sec. 4, before applying the proposed methods as a recommender system for images in Sec. 4.1.

### 1.1 Previous Work

In statistics, modelling data from related scenarios is typically done via mixed effects models or hierarchical Bayesian (HB) modelling [6]. In HB, parameters of models for individual scenarios (e.g. users in recommender systems) are assumed to be drawn from a common (hyper)prior distribution, allowing the individual models to interact and regularize each other. Recent examples of HB modelling in machine learning include [1, 2]. In other contexts, this learning framework is called multi-task learning [4]. Multi-task learning with Gaussian processes has been suggested by [8], yet with the rather stringent assumption that one has observations on the same set of points in each individual scenario. Based on sparse approximations of GPs, a more general GP multi-task learner with parametric covariance functions has been presented in [7]. In contrast, the approach presented in this paper only considers covariance *matrices* (and is thus non-parametric) in the first step. Only in a second extrapolation step, kernel smoothing leads to predictions based on a covariance function that is a data-driven combination of simple kernel functions.

## 2 Learning GP Kernel Matrices via EM

The learning task we are concerned with can be stated as follows: The data are observations from $M$ different scenarios. In the $i$.th scenario, we have observations $\boldsymbol{y}^i = (y^i_1, \ldots, y^i_{N^i})$ on a total of $N^i$ points, $X^i = \{\boldsymbol{x}^i_1, \ldots, \boldsymbol{x}^i_{N^i}\}$. In order to analyze this data in a hierarchical Bayesian way, we assume that the data for each scenario is a noisy sample of a Gaussian process (GP) with unknown mean and covariance function. We assume that mean and covariance function are shared across different scenarios.[1]

In the first modelling step presented in this section, we consider transductive learning ("labelling a partially labelled data set"), that is, we are interested in the model's behavior only on points $X$, with $X = \bigcup_{i=1}^M X^i$ and cardinality $N = |X|$. This situation is relevant for most collaborative filtering applications. Thus, test points are the unlabelled points in each scenario. This reduces the whole "infinite dimensional" Gaussian process to its finite dimensional projection on points $X$, which is an $N$-variate Gaussian distribution with covariance matrix $K$ and mean vector $\boldsymbol{m}$. For the EM algorithm to work, we also require that there is some overlap between scenarios, that is, $X^i \cap X^j \neq \emptyset$ for some $i, j$. Coming back to the user modelling problem mentioned above, this means that at least some items have been rated by more than one user.

Thus, our first modelling step focusses on directly learning the covariance matrix $K$ and

$m$ from the data via an efficient EM algorithm. This may be of particular help in problems where one would need to specify a complex (e.g. nonstationary) covariance function.

Following the hierarchical Bayesian assumption, the data observed in each scenario is thus a partial sample from $\mathcal{N}(\boldsymbol{y}\,|\,\boldsymbol{m}, K + \sigma^2 \mathbf{1})$, where $\mathbf{1}$ denotes the unit matrix. The joint model is simply

$$p(\boldsymbol{m}, K) \prod_{i=1}^{M} p(\boldsymbol{y}^i\,|\,\boldsymbol{f}^i) p(\boldsymbol{f}^i\,|\,\boldsymbol{m}, K), \tag{1}$$

where $p(\boldsymbol{m}, K)$ denotes the prior distribution for mean and covariance. We assume a Gaussian likelihood $p(\boldsymbol{y}^i\,|\,\boldsymbol{f}^i)$ with diagonal covariance matrix $\sigma^2 \mathbf{1}$.

## 2.1 EM Learning

For the above hierarchical Bayesian model, Eq. (1), the marginal likelihood becomes

$$p(\boldsymbol{m}, K) \prod_{i=1}^{M} \int p(\boldsymbol{y}^i\,|\,\boldsymbol{f}^i) p(\boldsymbol{f}^i\,|\,\boldsymbol{m}, K)\, d\boldsymbol{f}^i. \tag{2}$$

To obtain simple and stable solutions when estimating $\boldsymbol{m}$ and $K$ from the data, we consider point estimates of the parameters $\boldsymbol{m}$ and $K$, based on a penalized likelihood approach with conjugate priors.[2] The conjugate prior for mean $\boldsymbol{m}$ and covariance $K$ of a multivariate Gaussian is the so-called Normal-Wishart distribution [6], which decomposes into the product of an inverse Wishart distribution for $K$ and a Normal distribution for $\boldsymbol{m}$,

$$p(\boldsymbol{m}, K) = \mathcal{N}(\boldsymbol{m}\,|\,\boldsymbol{\nu}, \eta^{-1}K) \mathrm{Wi}^{-1}(K|\alpha, U). \tag{3}$$

That is, the prior for the Gram matrix $K$ is given by an inverse Wishart distribution with scalar parameter $\alpha > 1/2(N-1)$ and $U$ being a symmetric positive-definite matrix. Given the covariance matrix $K$, $\boldsymbol{m}$ is Gaussian distributed with mean $\boldsymbol{\nu}$ and covariance $\eta^{-1}K$, where $\eta$ is a positive scalar. The parameters can be interpreted in terms of an equivalent data set for the mean (this data set has size $A$, with $A = \nu$, and mean $\boldsymbol{\mu} = \boldsymbol{\nu}$) and a data set for the covariance that has size $B$, with $\alpha = (B + N)/2$, and covariance $S$, $U = (B/2)S$.

In order to write down the EM algorithm in a compact way, we denote by $\mathrm{I}(i)$ the set of indices of those data points that have been observed in the $i$.th scenario, that is $\mathrm{I}(i) = \{j\,|\,j \in \{1, \dots, N\}$ and $\boldsymbol{x}_j \in X^i\}$. Keep in mind that in most applications of interest $N^i \ll N$ such that most targets are missing in training. $K_{\mathrm{I}(i),\mathrm{I}(i)}$ denotes the square submatrix of $K$ that corresponds to points $\mathrm{I}(i)$, that is, the covariance matrix for points in the $i$.th scenario. By $K_{\cdot,\mathrm{I}(i)}$ we denote the covariance matrix of all $N$ points versus those in the $i$.th scenario.

### 2.1.1 E-step

In the E-step, one first computes $\tilde{\boldsymbol{f}}^i$, the expected value of functional values on all $N$ points for each scenario $i$. The expected value is given by the standard equations for the predictive mean of Gaussian process models, where the covariance functions are replaced by corresponding sub-matrices of the current estimate for $K$:

$$\tilde{\boldsymbol{f}}^i = K_{\cdot,\mathrm{I}(i)}(K_{\mathrm{I}(i),\mathrm{I}(i)} + \sigma^2 \mathbf{1})^{-1}(\boldsymbol{y}^i - \boldsymbol{m}_{\mathrm{I}(i)}) + \boldsymbol{m}, \quad i = 1, \dots, M. \tag{4}$$

Also, covariances between all pairs of points are estimated, based on the predictive covariance for the GP models: ($^\top$ denotes matrix transpose)

$$\tilde{C}^i = K - K_{\cdot,\mathrm{I}(i)}(K_{\mathrm{I}(i),\mathrm{I}(i)} + \sigma^2 \mathbf{1})^{-1} K_{\cdot,\mathrm{I}(i)}^\top, \quad i = 1, \dots, M. \tag{5}$$

### 2.1.2 M-step

In the M-step, the vector of mean values $\boldsymbol{m}$, the covariance matrix $K$ and the noise variance $\sigma^2$ are being updated. Denoting the updated quantities by $\boldsymbol{m}'$, $K'$, and $(\sigma^2)'$, we get

$$\boldsymbol{m}' = \frac{1}{M+A}\left(A\boldsymbol{\mu} + \sum_{i=1}^{M}\tilde{\boldsymbol{f}}^i\right)$$

$$K' = \frac{1}{M+B}\left(A(\boldsymbol{m}'-\boldsymbol{\mu})(\boldsymbol{m}'-\boldsymbol{\mu})^\top + BS + \sum_{i=1}^{M}\left((\tilde{\boldsymbol{f}}^i-\boldsymbol{m}')(\tilde{\boldsymbol{f}}^i-\boldsymbol{m}')^\top + \tilde{C}^i\right)\right)$$

$$(\sigma^2)' = \frac{1}{N}\left(\sum_{i=1}^{M}\|\boldsymbol{y}^i - \tilde{\boldsymbol{f}}^i_{\mathrm{I}(i)}\|^2 + \mathrm{trace}\,\tilde{C}^i_{\mathrm{I}(i),\mathrm{I}(i)}\right).$$

An intuitive explanation of the M-step is as follows: The new mean $\boldsymbol{m}'$ is a weighted combination of the prior mean, weighted by the equivalent sample size, and the predictive mean. The covariance update is a sum of four terms. The first term is typically irrelevant, it is a result of the coupling of the Gaussian and the inverse Wishart prior distributions via $K$. The second term contains the prior covariance matrix, again weighted by the equivalent sample size. As the third term, we get the empirical covariance, based on the estimated and measured functional values $\boldsymbol{f}^i$. Finally, the fourth term gives a correction term to compensate for the fact that the functional values $\boldsymbol{f}^i$ are only estimates, thus the empirical covariance will be too small.

## 3   Learning the Covariance Function via Generalized Nyström

Using the EM algorithm described in Sec. 2.1, one can easily and efficiently learn a covariance matrix $K$ and mean vector $\boldsymbol{m}$ from data obtained in different related scenarios. Once $K$ is found, predictions within the set $X$ can easily be made, by appealing to the same equations used in the EM algorithm (Eq. (4) for the predictive mean and Eq. (5) for the covariance). This would, for example, be of interest in a collaborative filtering application with a fixed set of items. In this section we describe how the covariance can be generalized to new inputs $\boldsymbol{z} \notin X$.

Note that, in all of the EM algorithm, the content features $\boldsymbol{x}^i_j$ do not contribute at all. In order to generalize the learned covariance matrix, we employ a kernel smoother with an auxiliary kernel function $r(\cdot,\cdot)$ that takes a pair of content features as input. As a constraint, we need to guarantee that the derived kernel is positive definite, such that straightforward interpolation schemes cannot readily be applied. Thus our strategy is to interpolate the eigenvectors of $K$ instead and subsequently derive a positive definite kernel. This approach is related to the Nyström method, which is primarily a method for extrapolating eigenfunctions that are only known at a discrete set of points. In contrast to Nyström, the extrapolating smoothing kernel is not known in our setting and we employ a generic smoothing kernel $r(\cdot,\cdot)$ instead [12].

Let $K = U\Lambda U^T$ be the eigendecomposition of covariance matrix $K$, with a diagonal matrix of eigenvalues $\Lambda$ and orthonormal eigenvectors $U$. With $V = U\Lambda^{1/2}$, the columns of $V$ are scaled eigenvectors. We now approximate the $i$-th scaled eigenvector $\boldsymbol{v}_i$ by a Gaussian process with covariance function $r(\cdot,\cdot)$ and obtain as an approximation of the scaled eigenfunction

$$\phi_i(\boldsymbol{w}) = \sum_{j=1}^{N} r(\boldsymbol{w},\boldsymbol{x}_j)b_{i,j} \tag{6}$$

with weights $\boldsymbol{b}_i = (b_{i,1}, \ldots, b_{i,N})^\top = (R + \lambda I)^{-1}\boldsymbol{v}_i$. $R$ denotes the Gram matrix for the smoothing kernel on all $N$ points. An additional regularization term $\lambda I$ is introduced to stabilize the inverse. Based on the approximate scaled eigenfunctions, the resulting kernel function is simply

$$l(\boldsymbol{w}, \boldsymbol{z}) = \sum_i \phi_i(\boldsymbol{w})\phi_i(\boldsymbol{z}) = \boldsymbol{r}(\boldsymbol{w})^\top (R + \lambda I)^{-1} K (R + \lambda I)^{-1} \boldsymbol{r}(\boldsymbol{z}). \tag{7}$$

with $\boldsymbol{r}(\boldsymbol{w})^\top = (r(\boldsymbol{x}_1, \boldsymbol{w}), \ldots, r(\boldsymbol{x}_N, \boldsymbol{w}))$. $R$ (resp. $L$) are the Gram matrices at the training data points $X$ for kernel function $r$ (resp. $l$) . $\lambda$ is a tuning parameter that determines which proportion of $K$ is explained by the content kernel. With $\lambda = 0$, $L = K$ is reproduced which means that all of $K$ can be explained by the content kernel. With $\lambda \to \infty$ then $l(\boldsymbol{w}, \boldsymbol{z}) \to 0$ and no portion of $K$ is explained by the content kernel.[3] Also, note that the eigenvectors are only required in the derivation, and do not need to be calculated when evaluating the kernel.[4]

Similarly, one can build a kernel smoother to extrapolate from the mean vector $\boldsymbol{m}$ to an approximate mean function $\hat{m}(\cdot)$. The prediction for a new object $\boldsymbol{v}$ in scenario $i$ thus becomes

$$f^i(v) = \hat{m}(v) + \sum_{j \in I(i)} l(v, x_j)\,\beta_j^i \tag{8}$$

with weights $\boldsymbol{\beta}$ given by $\boldsymbol{\beta}^i = (K_{I(i),I(i)} + \sigma^2 I)^{-1}(\boldsymbol{y}^i - m_{I(i)})$.

It is important to note $l$ has a much richer structure than the auxiliary kernel $r$. By expanding the expression for $l$, one can see that $l$ amounts to a data-dependent covariance function that can be written as a superposition of kernels $r$,

$$l(v, w) = \sum_{i=1}^{N} r(x_i, v)a_j^w, \tag{9}$$

with input dependent weights $\boldsymbol{a}^w = (R + \lambda I)^{-1} K (R + \lambda I)^{-1} \boldsymbol{r}_w$.

## 4  Experiments

We first illustrate the process of covariance matrix learning on a small toy example: Data is generated by sampling from a Gaussian process with the nonstationary "neural network covariance function" [11]. Independent Gaussian noise of variance $10^{-4}$ is added. Input points $X$ are 100 randomly placed points in the interval $[-1, 1]$. We consider $M = 20$ scenarios, where each scenario has observations on a random subset $X^i$ of $X$, with $N^i \approx 0.1N$. In Fig. 1(a), each scenario corresponds to one "noisy line" of points.

Using the EM-based covariance matrix learning (Sec. 2.1) on this data, the nonstationarity of the data does no longer pose problems, as Fig. 1 illustrates. The (stationary) covariance matrix shown in Fig. 1(c) was used both as the initial value for $K$ and for the prior covariance $S$ in Eq. (3). While the learned covariance matrix Fig. 1(d) does not fully match the true covariance, it clearly captures the nonstationary effects.

### 4.1  A Recommendation Engine

As a testbed for the proposed methods, we consider an information filtering task. The goal is to predict individual users' preferences for a large collection of art images[5], where

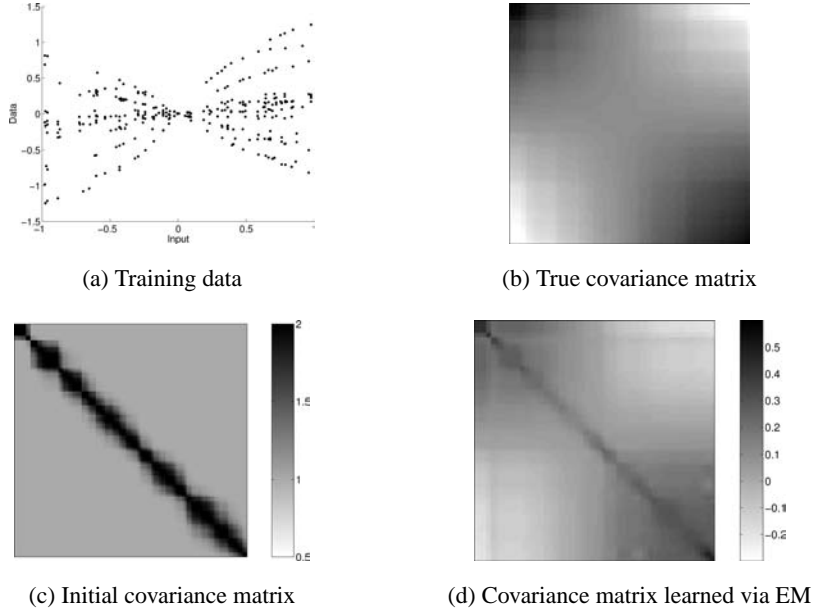

(a) Training data

(b) True covariance matrix

(c) Initial covariance matrix

(d) Covariance matrix learned via EM

Figure 1: Example to illustrate covariance matrix learning via EM. The data shown in (a) was drawn from a Gaussian process with a nonstationary "neural network" covariance function. When initialized with the stationary matrix shown in (c), EM learning resulted in the covariance matrix shown in (d). Comparing the learned matrix (d) with the true matrix (b) shows that the nonstationary structure is captured well

each user rated a random subset out of a total of 642 paintings, with ratings "like" $(+1)$, "dislike"$(-1)$, or "not sure" $(0)$. In total, ratings from $M = 190$ users were collected, where each user had rated 89 paintings on average. Each image is also described by a 275-dimensional feature vector (containing correlogram, color moments, and wavelet texture).

Fig. 2(a) shows ROC curves for collaborative filtering when preferences of unrated items within the set of 642 images are predicted. Here, our transductive approach (Eq. (4), "GP with EM covariance") is compared with a collaborative approach using Pearson correlation [3] ("Collaborative Filtering") and an alternative nonparametric hierarchical Bayesian approach [13] ("Hybrid Filter"). All algorithms are evaluated in a 10-fold cross validation scheme (repeated 10 times), where we assume that ratings for 20 items are known for each test user. Based on the 20 known ratings, predictions can be made for all unrated items. We obtain an ROC curve by computing sensitivity and specificity for the proportion of truly liked paintings among the $N$ top ranked paintings, averaged over $N$. The figure shows that our approach is considerably better than collaborative filtering with Pearson correlation and even gains a (yet small) advantage over the hybrid filtering technique.

Note that the EM algorithm converged[6] very quickly, requiring about 4–6 EM steps to learn the covariance matrix $K$. Also, we found that the performance is rather insensitive with respect to the hyperparameters, that is, the choice of $\mu$, $S$ and the equivalent sample sizes $A$ and $B$.

Fig. 2(b) shows ROC curves for the inductive setting where predictions for items outside

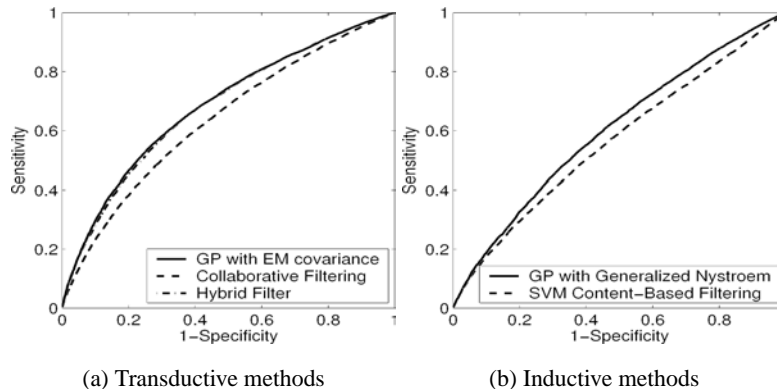

(a) Transductive methods          (b) Inductive methods

Figure 2: ROC curves of different methods for predicting user preferences for art images

the training set are to be made (sometimes referred to as the "new item problem"). Shown is the performance obtained with the generalized Nyström method ( Eq. (8), "GP with Generalized Nyström")[7], and when predicting user preferences from image features via an SVM with squared exponential kernel ("SVM content-based filtering"). It is apparent that the new approach with the learned kernel is superior to the standard SVM approach. Still, the overall performance of the inductive approach is quite limited. The low-level content features are only very poor indicators for the high level concept "liking an art image", and inductive approaches in general need to rely on content-dependent collaborative filtering. The purely content-independent collaborative effect, which is exploited in the transductive setting, cannot be generalized to new items. The purely content-independent collaborative effect can be viewed as correlated noise in our model.

## 5   Summary and Conclusions

This article introduced a novel method of learning Gaussian process covariance functions from multi-task learning problems, using a hierarchical Bayesian framework. In the hierarchical framework, the GP models for individual scenarios borrow strength from each other via a common prior for mean and covariance. The learning task was solved in two steps: First, an EM algorithm was used to learn the shared mean vector and covariance matrix on a fixed set of points. In a second step, the learned covariance matrix was generalized to new points via a generalized form of Nyström method. Our initial experiments, where we use the method as a recommender system for art images, showed very promising results. Also, in our approach, a clear distinction is made between content-dependent and content-independent collaborative filtering.

We expect that our approach will be even more effective in applications where the content features are more powerful (e.g. in recommender systems for textual items such as news articles), and allow a even better prediction of user preferences.

**Acknowledgements** This work was supported in part by the IST Programme of the European Union, under the PASCAL Network of Excellence (EU # 506778).

## Footnotes

[1]Alternative HB approaches for collaborative filtering, like that discussed in [5], assume that *model weights* are drawn from a shared Gaussian distribution.

[2]An efficient EM-based solution for the case $\sigma^2 = 0$ is also given by [9].

[3]Note that, also if the true interpolating kernel was known, i.e., $r = k$, and with $\lambda = 0$, we obtain $l(\boldsymbol{w}, \boldsymbol{z}) = k(\boldsymbol{w}, \boldsymbol{z})K^{-1}k(\boldsymbol{w}, \boldsymbol{z})$ which is the approximate kernel obtained with Nyström.

[4]A related form of kernel matrix extrapolation has been recently proposed by [10].

[5]http://honolulu.dbs.informatik.uni-muenchen.de:8080/paintings/index.jsp

[6]$S$ was set by learning a standard parametric GPR model from the preference data of one randomly chosen user, setting kernel parameters via marginal likelihood, and using this model to generate a full covariance matrix for all points.

[7]To obtain the kernel $r$, we fitted GP user preference models for a few randomly chosen users, with individual ARD weights for each input dimension in a squared exponential kernel. ARD weights for $r$ are taken to be the medians of the fitted ARD weights.

## References

[1] Bakker, B. and Heskes, T. Task clustering and gating for bayesian multitask learning. *Journal of Machine Learning Research*, 4:83–99, 2003.

[2] Blei, D. M., Ng, A. Y., and Jordan, M. I. Latent Dirichlet allocation. *Journal of Machine Learning Research*, 3:993–1022, 2003.

[3] Breese, J. S., Heckerman, D., and Kadie, C. Empirical analysis of predictive algorithms for collaborative filtering. Tech. Rep. MSR-TR-98-12, Microsoft Research, 1998.

[4] Caruana, R. Multitask learning. *Machine Learning*, 28(1):41–75, 1997.

[5] Chapelle, O. and Harchaoui, Z. A machine learning approach to conjoint analysis. In L. Saul, Y. Weiss, and L. Bottou, eds., *Neural Information Processing Systems 17*. MIT Press, 2005.

[6] Gelman, A., Carlin, J., Stern, H., and Rubin, D. *Bayesian Data Analysis*. CRCPress, 1995.

[7] Lawrence, N. D. and Platt, J. C. Learning to learn with the informative vector machine. In R. Greiner and D. Schuurmans, eds., *Proceedings of ICML04*. Morgan Kaufmann, 2004.

[8] Minka, T. P. and Picard, R. W. Learning how to learn is learning with point sets, 1999. Unpublished manuscript. Revised 1999.

[9] Schafer, J. L. *Analysis of Incomplete Multivariate Data*. Chapman&Hall, 1997.

[10] Vishwanathan, S., Guttman, O., Borgwardt, K. M., and Smola, A. Kernel extrapolation, 2005. Unpublished manuscript.

[11] Williams, C. K. Computation with infinite neural networks. *Neural Computation*, 10(5):1203–1216, 1998.

[12] Williams, C. K. I. and Seeger, M. Using the nyström method to speed up kernel machines. In T. K. Leen, T. G. Dietterich, and V. Tresp, eds., *Advances in Neural Information Processing Systems 13*, pp. 682–688. MIT Press, 2001.

[13] Yu, K., Schwaighofer, A., Tresp, V., Ma, W.-Y., and Zhang, H. Collaborative ensemble learning: Combining collaborative and content-based information filtering via hierarchical Bayes. In C. Meek and U. Kjærulff, eds., *Proceedings of UAI 2003*, pp. 616–623, 2003.

[14] Zhu, X., Ghahramani, Z., and Lafferty, J. Semi-supervised learning using Gaussian fields and harmonic functions. In *Proceedings of ICML03*. Morgan Kaufmann, 2003.

## Appendix

To derive an EM algorithm for Eq. (2), we treat the functional values $\boldsymbol{f}^i$ in each scenario $i$ as the unknown variables. In each EM iteration $t$, the parameters to be estimated are $\theta^{(t)} = \{\boldsymbol{m}^{(t)}, K^{(t)}, \sigma^{2(t)}\}$. In the E-step, the sufficient statistics are computed,

$$E\big(\sum_{i=1}^{M} \boldsymbol{f}^i \,|\, \boldsymbol{y}^i, \theta^{(t)}\big) = \sum_{i=1}^{M} \tilde{\boldsymbol{f}}^{i,(t)} \tag{10}$$

$$E\big(\sum_{i=1}^{M} \boldsymbol{f}^i (\boldsymbol{f}^i)^\top \,|\, \boldsymbol{y}^i, \theta^{(t)}\big) = \sum_{i=1}^{M} \Big(\tilde{\boldsymbol{f}}^{i,(t)} (\tilde{\boldsymbol{f}}^{i,(t)})^\top + \tilde{C}^i\Big) \tag{11}$$

with $\tilde{\boldsymbol{f}}^i$ and $\tilde{C}^i$ defined in Eq. (4) and (5). In the M-step, the parameters $\theta$ are re-estimated as $\theta^{(t+1)} = \arg\max_\theta Q(\theta \,|\, \theta^{(t)})$, with

$$Q(\theta \,|\, \theta^{(t)}) = \mathrm{E}\left[l_p(\theta \,|\, \boldsymbol{f}, \boldsymbol{y}) \,|\, \boldsymbol{y}, \theta^{(t)}\right], \tag{12}$$

where $l_p$ stands for the penalized log-likelihood of the complete data,

$$l_p(\theta \,|\, \boldsymbol{f}, \boldsymbol{y}) = \log \mathrm{Wi}^{-1}(K \,|\, \alpha, \beta) + \log \mathcal{N}(\boldsymbol{m} \,|\, \nu, \eta^{-1} K) +$$
$$+ \sum_{i=1}^{M} \log \mathcal{N}(\tilde{\boldsymbol{f}}^i \,|\, \boldsymbol{m}, K) + \sum_{i=1}^{M} \log \mathcal{N}(\boldsymbol{y}_{\mathrm{I}(i)}^i \,|\, \tilde{\boldsymbol{f}}_{\mathrm{I}(i)}^i, \sigma^2 \mathbf{1}) \tag{13}$$

Updated parameters are obtained by setting the partial derivatives of $Q(\theta \,|\, \theta^{(t)})$ to zero.